# Structure Driven Image Database Retrieval

**Jeremy S. De Bonet & Paul Viola**
Artificial Intelligence Laboratory
Learning & Vision Group
545 Technology Square Massachusetts Institute of Technology
. Cambridge, MA 02139

EMAIL: jsd@ai.mit.edu & viola@ai.mit.edu
HOMEPAGE: http://www.ai.mit.edu/projects/lv

## Abstract

A new algorithm is presented which approximates the perceived visual similarity between images. The images are initially transformed into a feature space which captures visual structure, texture and color using a tree of filters. Similarity is the inverse of the distance in this *perceptual feature space*. Using this algorithm we have constructed an image database system which can perform example based retrieval on large image databases. Using carefully constructed target sets, which limit variation to only a single visual characteristic, retrieval rates are quantitatively compared to those of standard methods.

## 1 Introduction

Without supplementary information, there exists no way to directly measure the similarity between the content of images. In general, one cannot answer a question of the form: "is image A more like image B or image C?" without defining the criteria by which this comparison is to be made. People perform such tasks by inferring some criterion, based on their visual experience or by complex reasoning about the situations depicted in the images. Humans are very capable database searchers. They can perform simple searches like, "find me images of cars", or more complex or loosely defined searches like, "find me images that depict pride in America". In either case one must examine all, or a large portion, of the database. As the prevalence and size of multimedia databases increases, automated techniques will become critical in the successful retrieval of relevant information. Such techniques must be able to measure the similarity between the visual content of natural images.

Many algorithms have been proposed for image database retrieval. For the most part these techniques compute a feature vector from an image which is made up of a handful of image measurements. Visual or semantic distance is then equated with feature distance. Examples include color histograms, texture histograms, shape boundary descriptors, eigenimages, and hybrid schemes (QBIC, ; Niblack et al., 1993; Virage, ; Kelly, Cannon and Hush, 1995; Pentland, Picard and Sclaroff, 1995; Picard and Kabir, 1993; Santini and Jain, 1996). A query to such a system typically consists of specifying two types of parameters: the target values of each of the measurements; and a set of weights, which determine the relative importance of deviations from the target in each measurement dimension. The features used by these systems each capture some very general property of images. As a result of their lack of specificity however, many images which are actually very different in content generate the same feature responses.

In contrast our approach extracts thousands of very specific features. These features measure both local texture and global structure. The feature extraction algorithm computes color, edge orientation, and other local properties at many resolutions. This sort of multi-scale feature analysis is of critical importance in visual recognition and has been used successfully in the context of object recognition (von der Malsburg, 1988; Rao and Ballard, 1995; Viola, 1996)

Our system differs from others because it detects not only first order relationships, such as edges or color, but also measures how these first order relationships are related to one another. Thus by finding patterns between image regions with particular local properties, more complex – and therefore more discriminating – features can be extracted. This type of repeated, non-linear, feature detection bears a strong resemblance to the response properties of visual cortex cells (Desimone et al., 1984). While the mechanism for the responses of these cells is not yet clear, this work supports the conclusion that this type of representation is very useful in practical visual processing.

## 2  Computing the Characteristic Signature

The "texture-of-texture" measurements are based on the outputs of a tree of non-linear filtering operations. Each path through the tree creates a particular filter network, which responds to certain structural organization in the image. Measuring the appropriately weighted difference between the signatures of images in the database and the set of query-images, produces a similarity measure which can be used to rank and sort the images in the database.

The computation of the characteristic signature is straightforward. At the highest level of resolution the image is convolved with a set of 25 local linear features including oriented edges and bars. The results of these convolutions are 25 feature response images. These images are then rectified by squaring, which extracts the *texture energy* in the image, and then downsampled by a factor of two. At this point there are 25 half scale output images which each measure a local textural property of the input image. For example one image is sensitive to vertical edges, and responds strongly to both skyscrapers and picket fences.

Convolution, rectification and downsampling is then repeated on each of these 25 half resolution images producing 625 quarter scale "texture-of-texture" images. The second layer will respond strongly to regions where the texture specified in the first layer has a particular spatial arrangement. For example if horizontal alignments of vertical texture are detected, there will be a strong response to a picket fence and little response to a skyscraper. With additional layers additional specificity is

achieved; repeating this procedure a third time yields 15,625 meta-texture images at eighth scale.

Each of the resulting meta-textures is then summed to compute a single value and provides one element in the characteristic signature. When three channels of color are included there are a total of 46,875 elements in the characteristic signature. Once computed, the signature elements are normalized to reduce the effects of contrast changes.

More formally the characteristic signature of an image is given by:

$$S_{i,j,k,c}(I) = \sum_{pixels} E_{i,j,k}(I_c) \tag{1}$$

where $I$ is the image, $i$, $j$ and $k$ index over the different types of linear filters, and $I_c$ are the different color channels of the image. The definition of $E$ is:

$$E_i(I) = 2 \downarrow [(F_i \otimes I)^2] \tag{2}$$

$$E_{i,j}(I) = 2 \downarrow [(F_j \otimes E_i(I))^2] \tag{3}$$

$$E_{i,j,k}(I) = 2 \downarrow [(F_k \otimes E_{i,j}(I))^2] . \tag{4}$$

where $F_i$ is the $i$th filter and $2 \downarrow$ is the downsampling operation.

## 3   Using Characteristic Signatures To Form Image Queries

In the "query by image" paradigm, we describe similarity in terms of the difference between an image and a group of example query images. This is done by comparing the characteristic signature of the image to the mean signature of the query images. The relative importance of each element of the characteristic signature in determining similarity is proportional to the inverse variance of that element across the example-image group:

$$L = -\sum_i \sum_j \sum_k \sum_c \frac{\left[\overline{S_{i,j,k,c}(I_q)} - S_{i,j,k,c}(I_{test})\right]^2}{Var\left[S_{i,j,k,c}(I_q)\right]} \tag{5}$$

where $\overline{S_{i,j,k,c}(I_q)}$ and $Var\left[S_{i,j,k,c}(I_q)\right]$ are the mean and variance of the characteristic signatures computed over the query set. This is a diagonal approximation of the Mahalanobis distance (Duda and Hart, 1973). It has the effect of normalizing the vector-space defined by the characteristic signatures, so that characteristic elements which are salient within the group of query images contribute more to the overall similarity of an image.

In Figure 1 three 2D projections of these 46,875 dimensional characteristic signature space are shown. The data points marked with circles are generated by the 10 images shown at the top of Figure 3. The remaining points are generated by 2900 distractor images. Comparing (a) and (b) we see that in some projections the images cluster tightly, while in others they are distributed. Given a sample of images from the target set we can observe the variation in each possible projection axis. Most of the time the axes shown in (a) will be strongly discounted by the algorithm because these features are not consistent across the query set. Similarly the axes from (b) will receive a large weight because the target images have very consistent values.

The axes along which target groups cluster, however, differ from target group to target group. As a result it is not possible to conclude that the axes in (b) are simply better than the axes in (a). In Figure 1 (c) the same projection is shown again this time with a different target set highlighted (with asterisks).

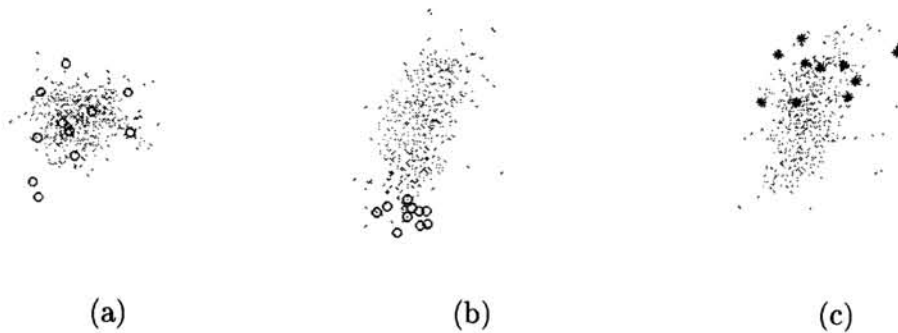

(a)            (b)            (c)

Figure 1: In some projections target groups do not cluster (a), and they do in others (b). However, different target groups will not necessarily cluster in the same projections (c).

## 4  Experiments

In the first set of experiments we use a database of 2900 images from 29 Corel Photo CD (collections 1000-2900.) Figure 2 shows the results of typical user query on this system. The top windows in each Figure contain the query-images submitted by the user. The bottom windows show the thirty images found to be most similar; similarity decreases from upper left (most similar) to lower right.

Though these examples provide an anecdotal indication that the system is generating similarity measures which roughly conform to human perception, it is difficult to fully characterize the performance of this image retrieval technique. This is a fundamental problem of the domain. Images vary from each other in an astronomical number of ways, and similarity is perceived by human observers based upon complex interactions between recognition, cognition, and assumption. It seems unlikely that an absolute criterion for image similarity can ever be determined, if one truly exists. However using sets of images which we believe are visually similar, we can establish a basis for comparing algorithms.

To better measure the performance of the system we added a set of 10 images to the 2900 image database and attempted to retrieve these new images. We compare the performance of the present system to ten other techniques. Though these techniques are not as sophisticated as those used in systems developed by other researchers, they are indicative of the types of methods which are prevalent in the literature.

In each experiment we measure the retrieval rates for a set of ten target images which we believe to be visually similar because they consist of images of a single scene. Images in the target set differ due to variation of a single visual characteristic. In some of the target sets the photographic conditions have been changed, either by moving the camera, the objects or the light. In other target sets post photograph image manipulation has been performed. Two example target sets are shown in Figure 3.

In each experiment we perform 45 database queries using every possible pair of images from the target set. The retrieval methods compared are: **ToT** The current textures-of-textures system; **RGB-216(or 512)C** R,G,B color histograms using 216 (or 512) bins by dividing each color dimension into 6 (or 8) regions. The target histogram is generated by combining the histograms from the two model images; **HSV-216(or 512)C** same using H,S,V color space; **RGB(or HSV)-216(or**

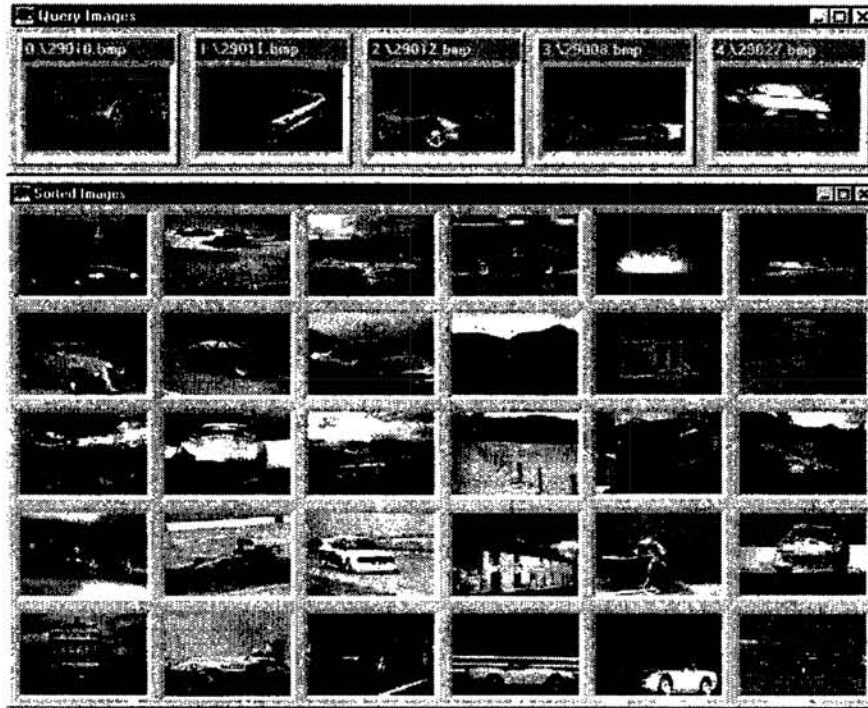

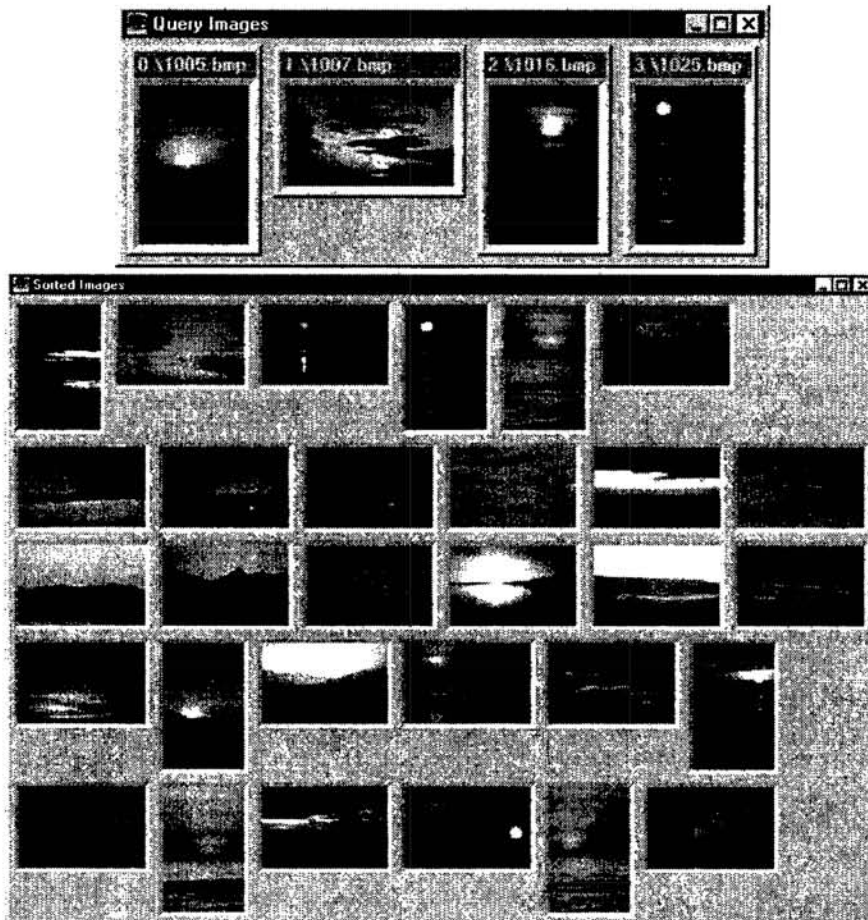

Figure 2: Sample queries and top 30 responses.

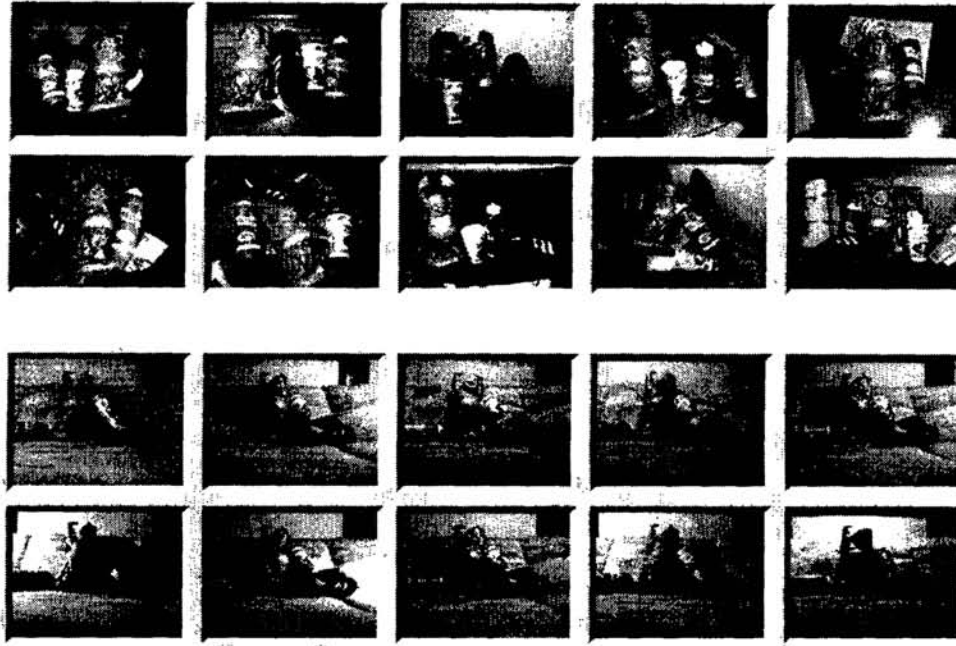

Figure 3: Two target sets used in the retrieval experiments. TOP: Variation of object location. BOTTOM: Variation of hard shadows.

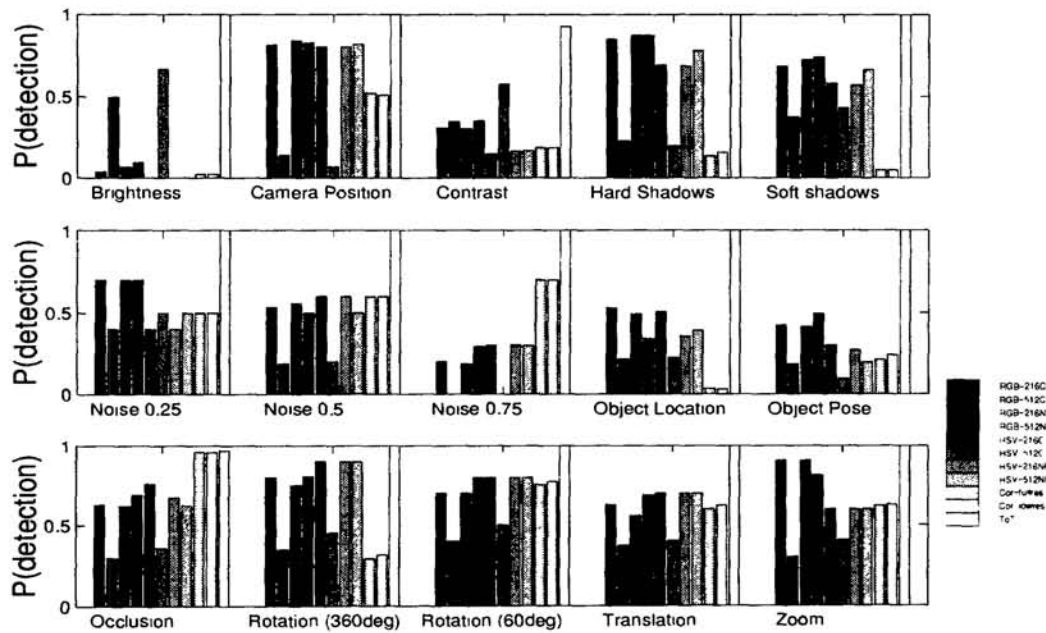

Figure 4: The percentage of target images ranked above all the distractor images, shown for 15 target sets such as those in Figure 3. The textures-of-textures model presented here achieves perfect performance in 13 of the 15 experiments.

**512)NN** histograms in which similarity is measured using a nearest neighbor metric; **COR-full(and low)res** full resolution (and 4× downsampled) image correlation.

Rankings for all 10 target images in each of the 45 queries are obtained for each variation. To get a comparative sense of the overall performance of each technique, we show the number of target images retrieved with a Neyman-Pearson criterion of zero, i.e. no false positives Figure 4. The textures-of-textures model substantially outperforms all of the other techniques, achieving perfect performance in 13 experiments.

## 5  Discussion

We have presented a technique for approximating perceived visual similarity, by measuring the structural content similarity between images. Using the high dimensional "characteristic signature" space representation, we directly compare database-images to a set of query-images. A world wide web interface to system has been created and is available via the URL:
                http://www.ai.mit.edu/~jsd/Research/ImageDatabase/Demo

Experiments indicate that the present system can retrieve images which share visual characteristics with the query-images, from a large non-homogeneous database. Further, it greatly outperforms many of the standard methods which form the basis of other systems.

Though the results presented here are encouraging, on real world queries, the retrieved images often contain many false alarms, such as those in Figure 2; however, we believe that with additional analysis performance can be improved.

## References

Desimone, R., Albright, T. D., Gross, C. G., and Bruce, C. (1984). Stimulus selective properties of inferior temporal neurons in the macaque. *Journal of Neuroscience*, 4:2051–2062.

Duda, R. and Hart, P. (1973). *Pattern Classification and Scene Analysis*. John Wiley and Sons.

Kelly, M., Cannon, T. M., and Hush, D. R. (1995). Query by image example: the candid approach. *SPIE Vol. 2420 Storage and Retrieval for Image and Video Databases III*, pages 238–248.

Niblack, V., Barber, R., Equitz, W., Flickner, M., Glasman, E., Petkovic, D., Yanker, P., Faloutsos, C., and Taubin, G. (1993). The qbic project: querying images by content using color, texture, and shape. *IS&T/SPIE 1993 International Symposium on Electronic Imaging: Science & Technology*, 1908:173–187.

Pentland, A., Picard, R. W., and Sclaroff, S. (1995). Photobook: Content-based manipulation of image databases. Technical Report 255, MIT Media Lab.

Picard, R. W. and Kabir, T. (1993). Finding similar patterns in large image databases. *ICASSP*, V:161–164.

QBIC. The ibm qbic project. Web: http://wwwqbic.almaden.ibm.com/.

Rao, R. P. N. and Ballard, D. (1995). Object indexing using an iconic sparse distributed memory. Technical Report TR-559, University of Rochester.

Santini, S. and Jain, R. (1996). Gabor space and the development of preattentive similarity. In *Proceedings of ICPR 96*. International Conference on Pattern Recognition, Vienna.

Viola, P. (1996). Complex feature recognition: A bayesian approach for learning to recognize objects. Technical Report 1591, MIT AI Lab.

Virage. The virage project. Web: http://www.virage.com/.

von der Malsburg, C. (1988). Pattern recognition by labeled graph matching. *Neural Networks*, 1:141–148.
